# Margin Maximizing Loss Functions

**Saharon Rosset**
Watson Research Center
IBM
Yorktown, NY, 10598
*srosset@us.ibm.com*

**Ji Zhu**
Department of Statistics
University of Michigan
Ann Arbor, MI, 48109
*jizhu@umich.edu*

**Trevor Hastie**
Department of Statistics
Stanford University
Stanford, CA, 94305
*hastie@stat.stanford.edu*

## Abstract

Margin maximizing properties play an important role in the analysis of classifi­cation models, such as boosting and support vector machines. Margin maximiza­tion is theoretically interesting because it facilitates generalization error analysis, and practically interesting because it presents a clear geometric interpretation of the models being built. We formulate and prove a sufficient condition for the solutions of regularized loss functions to converge to margin maximizing separa­tors, as the regularization vanishes. This condition covers the hinge loss of SVM, the exponential loss of AdaBoost and logistic regression loss. We also generalize it to multi-class classification problems, and present margin maximizing multi-class versions of logistic regression and support vector machines.

## 1 Introduction

Assume we have a classification "learning" sample $\{\mathbf{x}_i, y_i\}_{i=1}^n$ with $y_i \in \{-1, +1\}$. We wish to build a model $F(\mathbf{x})$ for this data by minimizing (exactly or approximately) a loss criterion $\sum_i C(y_i, F(\mathbf{x}_i)) = \sum_i C(y_i F(\mathbf{x}_i))$ which is a function of the *margins* $y_i F(\mathbf{x}_i)$ of this model on this data. Most common classification modeling approaches can be cast in this framework: logistic regression, support vector machines, boosting and more. The model $F(\mathbf{x})$ which these methods actually build is a *linear combination* of *dictionary func­tions* coming from a dictionary $\mathcal{H}$ which can be large or even infinite:

$$F(\mathbf{x}) = \sum_{h_j \in \mathcal{H}} \beta_j h_j(\mathbf{x})$$

and our prediction at point $\mathbf{x}$ based on this model is $\mathrm{sgn}F(\mathbf{x})$.

When $|\mathcal{H}|$ is large, as is the case in most boosting or kernel SVM applications, some regu­larization is needed to control the "complexity" of the model $F(\mathbf{x})$ and the resulting over-fitting. Thus, it is common that the quantity actually minimized on the data is a regularized version of the loss function:

$$\hat{\beta}(\lambda) = \min_{\beta} \sum_i C(y_i \beta' h(\mathbf{x}_i)) + \lambda \|\beta\|_p^p \tag{1}$$

where the second term penalizes for the $l_p$ norm of the coefficient vector $\beta$ ($p \geq 1$ for convexity, and in practice usually $p \in \{1, 2\}$), and $\lambda \geq 0$ is a tuning regularization parame­ter. The 1- and 2-norm support vector machine training problems with slack can be cast in this form ([6], chapter 12). In [8] we have shown that boosting approximately follows the

"path" of regularized solutions traced by (1) as the regularization parameter $\lambda$ varies, with the appropriate loss and an $l_1$ penalty.

The main question that we answer in this paper is: for what loss functions does $\hat{\beta}(\lambda)$ converge to an "optimal" separator as $\lambda \to 0$? The definition of "optimal" which we will use depends on the $l_p$ norm used for regularization, and we will term it the "$l_p$-margin maximizing separating hyper-plane". More concisely, we will investigate for which loss functions and under which conditions we have:

$$(2) \qquad \lim_{\lambda \to 0} \frac{\hat{\beta}(\lambda)}{\|\hat{\beta}(\lambda)\|} = \arg \max_{\|\beta\|_p = 1} \min_i y_i \beta' h(\mathbf{x}_i)$$

This margin maximizing property is interesting for three distinct reasons. First, it gives us a geometric interpretation of the "limiting" model as we relax the regularization. It tells us that this loss seeks to optimally separate the data by maximizing a distance between a separating hyper-plane and the "closest" points. A theorem by Mangasarian [7] allows us to interpret $l_p$ margin maximization as $l_q$ distance maximization, with $1/p + 1/q = 1$, and hence make a clear geometric interpretation. Second, from a learning theory perspective large margins are an important quantity — generalization error bounds that depend on the margins have been generated for support vector machines ([10] — using $l_2$ margins) and boosting ( [9] — using $l_1$ margins). Thus, showing that a loss function is "margin maximizing" in this sense is useful and promising information regarding this loss function's potential for generating good prediction models. Third, practical experience shows that exact or approximate margin maximizaion (such as non-regularized kernel SVM solutions, or "infinite" boosting) may actually lead to good classification prediction models. This is certainly not always the case, and we return to this hotly debated issue in our discussion.

Our main result is a sufficient condition on the loss function, which guarantees that (2) holds, *if the data is separable*, i.e. if the maximum on the RHS of (2) is positive. This condition is presented and proven in section 2. It covers the hinge loss of support vector machines, the logistic log-likelihood loss of logistic regression, and the exponential loss, most notably used in boosting. We discuss these and other examples in section 3. Our result generalizes elegantly to multi-class models and loss functions. We present the resulting margin-maximizing versions of SVMs and logistic regression in section 4.

## 2  Sufficient condition for margin maximization

The following theorem shows that if the loss function vanishes "quickly" enough, then it will be margin-maximizing as the regularization vanishes. It provides us with a unified margin-maximization theory, covering SVMs, logistic regression and boosting.

**Theorem 2.1** *Assume the data* $\{\mathbf{x}_i, y_i\}_{i=1}^n$ *is separable, i.e.* $\exists \beta$ *s.t.* $\min_i y_i \beta' h(\mathbf{x}_i) > 0$.
*Let* $C(y, f) = C(yf)$ *be a monotone non-increasing loss function depending on the margin only.*
*If* $\exists T > 0$ *(possibly* $T = \infty$ *) such that:*

$$(3) \qquad \lim_{t \nearrow T} \frac{C(t \cdot [1 - \epsilon])}{C(t)} = \infty, \ \forall \epsilon > 0$$

*Then* $C$ *is a margin maximizing loss function in the sense that* any *convergence point of the normalized solutions* $\frac{\hat{\beta}(\lambda)}{\|\hat{\beta}(\lambda)\|_p}$ *to the regularized problems (1) as* $\lambda \to 0$ *is an* $l_p$ *margin-maximizing separating hyper-plane. Consequently, if this margin-maximizing hyper-plane is* unique, *then the solutions converge to it:*

$$(4) \qquad \lim_{\lambda \to 0} \frac{\hat{\beta}(\lambda)}{\|\hat{\beta}(\lambda)\|_p} = \arg \max_{\|\beta\|_p = 1} \min_i y_i \beta' h(\mathbf{x}_i)$$

**Proof** We prove the result separately for $T = \infty$ and $T < \infty$.

**a.** $T = \infty$:

**Lemma 2.2** $\|\hat{\beta}(\lambda)\|_p \xrightarrow{\lambda \to 0} \infty$

**Proof** Since $T = \infty$ then $C(m) > 0 \; \forall m > 0$, and $\lim_{m \to \infty} C(m) = 0$. Therefore, for loss+penalty to vanish as $\lambda \to 0$, $\|\hat{\beta}(\lambda)\|_p$ must diverge, to allow the margins to diverge.

**Lemma 2.3** *Assume* $\beta_1, \beta_2$ *are two separating models, with* $\|\beta_1\|_p = \|\beta_2\|_p = 1$, *and* $\beta_1$ *separates the data better, i.e.:* $0 < m_2 = \min_i y_i h(\mathbf{x}_i)' \beta_2 < m_1 = \min_i y_i h(\mathbf{x}_i)' \beta_1$. *Then* $\exists U = U(m_1, m_2)$ *such that*

$$\forall t > U, \; \sum_i C(y_i h(\mathbf{x}_i)'(t\beta_1)) < \sum_i C(y_i h(\mathbf{x}_i)'(t\beta_2))$$

*In words, if* $\beta_1$ *separates better than* $\beta_2$ *then scaled-up versions of* $\beta_1$ *will incur smaller loss than scaled-up versions of* $\beta_2$, *if the scaling factor is large enough.*

**Proof** Since condition (3) holds with $T = \infty$, there exists $U$ such that $\forall t > U, \; \frac{C(tm_2)}{C(tm_1)} > n$. Thus from $C$ being non-increasing we immediately get:

$$\forall t > U, \; \sum_i C(y_i h(\mathbf{x}_i)'(t\beta_1)) \leq n \cdot C(tm_1) < C(tm_2) < \sum_i C(y_i h(\mathbf{x}_i)'(t\beta_2))$$

**Proof of case a.:** Assume $\beta^*$ is a convergence point of $\frac{\hat{\beta}(\lambda)}{\|\hat{\beta}(\lambda)\|_p}$ as $\lambda \to 0$, with $\|\beta^*\|_p = 1$.

Now assume by contradiction $\tilde{\beta}$ has $\|\tilde{\beta}\|_p = 1$ and bigger minimal $l_p$ margin. Denote the minimal margins for the two models by $m^*$ and $\tilde{m}$, respectively, with $m^* < \tilde{m}$.

By continuity of the minimal margin in $\beta$, there exists some open neighborhood of $\beta^*$ on the $l_p$ sphere:

$$N_{\beta^*} = \{\beta : \|\beta\|_p = 1, \; \|\beta - \beta^*\|_2 < \delta\}$$

and an $\epsilon > 0$, such that:

$$\min_i y_i \beta' h(\mathbf{x_i}) < \tilde{m} - \epsilon, \; \; \forall \beta \in N_{\beta^*}$$

Now by lemma 2.3 we get that exists $U = U(\tilde{m}, \tilde{m} - \epsilon)$ such that $t\tilde{\beta}$ incurs smaller loss than $t\beta$ for any $t > U$, $\beta \in N_{\beta^*}$. Therefore $\beta^*$ *cannot be a convergence point of* $\frac{\hat{\beta}(\lambda)}{\|\hat{\beta}(\lambda)\|_p}$.

**b.** $T < \infty$

**Lemma 2.4** $C(T) = 0$ *and* $C(T - \delta) > 0, \; \forall \delta > 0$.

**Proof** From condition (3), $\frac{C(T - T\epsilon)}{C(T)} = \infty$. Both results follow immediately, with $\delta = T\epsilon$.

**Lemma 2.5** $\lim_{\lambda \to 0} \min_i y_i \hat{\beta}(\lambda)' h(\mathbf{x}_i) = T$

**Proof** Assume by contradiction that there is a sequence $\lambda_1, \lambda_2, \ldots \searrow 0$ and $\epsilon > 0$ s.t. $\forall j, \; \min_i y_i \hat{\beta}(\lambda_j)' h(\mathbf{x}_i) \leq T - \epsilon$.
Pick any separating normalized model $\tilde{\beta}$ i.e. $\|\tilde{\beta}\|_p = 1$ and $\tilde{m} := \min_i y_i \tilde{\beta}' h(\mathbf{x}_i) > 0$. Then for *any* $\lambda < \tilde{m}^p \frac{C(T - \epsilon)}{T^p}$ we get:

$$\sum_i C(y_i \frac{T}{\tilde{m}} \tilde{\beta}' h(\mathbf{x}_i)) + \lambda \|\frac{T}{\tilde{m}} \tilde{\beta}\|_p^p < C(T - \epsilon)$$

since the first term (loss) is 0 and the penalty is smaller than $C(T - \epsilon)$ by condition on $\lambda$. But $\exists j_0$ s.t. $\lambda_{j_0} < \tilde{m}^p \frac{C(T-\epsilon)}{T^p}$ and so we get a contradiction to optimality of $\hat{\beta}(\lambda_{j_0})$, since we assumed $\min_i y_i \hat{\beta}(\lambda_{j_0})' h(\mathbf{x}_i) \leq T - \epsilon$ and thus:

$$\sum_i C(y_i \hat{\beta}(\lambda_{j_0})' h(\mathbf{x}_i)) \geq C(T - \epsilon)$$

We have thus proven that $\liminf_{\lambda \to 0} \min_i y_i \hat{\beta}(\lambda)' h(\mathbf{x}_i) \geq T$. It remains to prove equality. Assume by contradiction that for *some* value of $\lambda$ we have $m := \min_i y_i \hat{\beta}(\lambda)' h(\mathbf{x}_i) > T$. Then the re-scaled model $\frac{T}{m}\hat{\beta}(\lambda)$ has the same zero loss as $\hat{\beta}(\lambda)$, but a smaller penalty, since $\|\frac{T}{m}\hat{\beta}(\lambda)\| = \frac{T}{m}\|\hat{\beta}(\lambda)\| < \|\hat{\beta}(\lambda)\|$. So we get a contradiction to optimality of $\hat{\beta}(\lambda)$.

**Proof of case b.:** Assume $\beta^*$ is a convergence point of $\frac{\hat{\beta}(\lambda)}{\|\hat{\beta}(\lambda)\|_p}$ as $\lambda \to 0$, with $\|\beta^*\|_p = 1$. Now assume by contradiction $\tilde{\beta}$ has $\|\tilde{\beta}\|_p = 1$ and bigger minimal margin. Denote the minimal margins for the two models by $m^*$ and $\tilde{m}$, respectively, with $m^* < \tilde{m}$.

Let $\lambda_1, \lambda_2, ... \searrow 0$ be a sequence along which $\frac{\hat{\beta}(\lambda_j)}{\|\hat{\beta}(\lambda_j)\|_p} \to \beta^*$. By lemma 2.5 and our assumption, $\|\hat{\beta}(\lambda_j)\|_p \to \frac{T}{m^*} > \frac{T}{\tilde{m}}$. Thus, $\exists j_0$ such that $\forall j > j_0$, $\|\hat{\beta}(\lambda_j)\|_p > \frac{T}{\tilde{m}}$ and consequently:

$$\sum_i C(y_i \hat{\beta}(\lambda_j)' h(\mathbf{x}_i)) + \lambda\|\hat{\beta}(\lambda_j)\|_p^p > \lambda(\frac{T}{\tilde{m}})^p = \sum_i C(y_i \frac{T}{\tilde{m}}\tilde{\beta} h(\mathbf{x}_i)) + \lambda\|\frac{T}{\tilde{m}}\tilde{\beta}\|_p^p$$

So we get a contradiction to optimality of $\hat{\beta}(\lambda_j)$.

Thus we conclude for both cases **a.** and **b.** that any convergence point of $\frac{\hat{\beta}(\lambda)}{\|\hat{\beta}(\lambda)\|_p}$ must maximize the $l_p$ margin. Since $\|\frac{\hat{\beta}(\lambda)}{\|\hat{\beta}(\lambda)\|_p}\|_p = 1$, such convergence points obviously exist. If the $l_p$-margin-maximizing separating hyper-plane is unique, then we can conclude:

$$\frac{\hat{\beta}(\lambda)}{\|\hat{\beta}(\lambda)\|_p} \to \hat{\beta} := \arg \max_{\|\beta\|_p = 1} \min_i y_i \beta' h(\mathbf{x}_i)$$

**Necessity results**

A necessity result for margin maximization *on any separable data* seems to require either additional assumptions on the loss or a relaxation of condition (3). We conjecture that if we also require that the loss is convex and vanishing (i.e. $lim_{m \to \infty} C(m) = 0$) then condition (3) is sufficient *and necessary*. However this is still a subject for future research.

## 3   Examples

**Support vector machines**

Support vector machines (linear or kernel) can be described as a regularized problem:

$$(5) \qquad \min_\beta \sum_i [1 - y_i \beta' h(\mathbf{x}_i)]_+ + \lambda\|\beta\|_p^p$$

where $p = 2$ for the standard ("2-norm") SVM and $p = 1$ for the 1-norm SVM. This formulation is equivalent to the better known "norm minimization" SVM formulation in the sense that they have the same set of solutions as the regularization parameter $\lambda$ varies in (5) or the slack bound varies in the norm minimization formulation.

The loss in (5) is termed "hinge loss" since it's linear for margins less than 1, then fixed at 0 (see figure 1). The theorem obviously holds for $T = 1$, and it verifies our knowledge that the non-regularized SVM solution, which is the limit of the regularized solutions, maximizes the appropriate margin (Euclidean for standard SVM, $l_1$ for 1-norm SVM).

Note that our theorem indicates that the squared hinge loss (AKA truncated squared loss):

$$C(y_i, F(\mathbf{x}_i)) = [1 - y_i F(\mathbf{x}_i)]_+^2$$

is also a margin-maximizing loss.

**Logistic regression and boosting**

The two loss functions we consider in this context are:

(6) $\qquad\qquad\quad Exponential : \quad C_e(m) = \exp(-m)$

(7) $\qquad\qquad\quad Log\ likelihood : \quad C_l(m) = \log(1 + \exp(-m))$

These two loss functions are of great interest in the context of two class classification: $C_l$ is used in logistic regression and more recently for boosting [4], while $C_e$ is the implicit loss function used by AdaBoost - the original and most famous boosting algorithm [3] .

In [8] we showed that boosting approximately follows the regularized path of solutions $\hat{\beta}(\lambda)$ using these loss functions and $l_1$ regularization. We also proved that the two loss functions are very similar for positive margins, and that their regularized solutions converge to margin-maximizing separators. Theorem 2.1 provides a new proof of this result, since the theorem's condition holds with $T = \infty$ for both loss functions.

**Some interesting non-examples**

Commonly used classification loss functions which are not margin-maximizing include *any* polynomial loss function: $C(m) = \frac{1}{m}$, $C(m) = m^2$, etc. do not guarantee convergence of regularized solutions to margin maximizing solutions.

Another interesting method in this context is linear discriminant analysis. Although it does not correspond to the loss+penalty formulation we have described, it does find a "decision hyper-plane" in the predictor space.

For both polynomial loss functions and linear discriminant analysis it is easy to find examples which show that they are not necessarily margin maximizing on separable data.

## 4   A multi-class generalization

Our main result can be elegantly extended to versions of multi-class logistic regression and support vector machines, as follows. Assume the response is now multi-class, with $K \geq 2$ possible values i.e. $y_i \in \{c_1, ..., c_K\}$. Our model consists of a "prediction" for each class:

$$F_k(\mathbf{x}) = \sum_{h_j \in \mathcal{H}} \beta_j^{(k)} h_j(\mathbf{x})$$

with the obvious prediction rule at $\mathbf{x}$ being $\arg\max_k F_k(\mathbf{x})$.
This gives rise to a $K - 1$ dimensional "margin" for each observation. For $y = c_k$, define the margin vector as:

(8) $\qquad \mathbf{m}(c_k, f_1, ..., f_K) = (f_k - f_1, ..., f_k - f_{k-1}, f_k - f_{k+1}, ..., f_k - f_K)'$

And our loss is a function of this $K - 1$ dimensional margin:

$$C(y, f_1, ..., f_K) = \sum_k I\{y = c_k\} C(\mathbf{m}(c_k, f_1, ..., f_K))$$

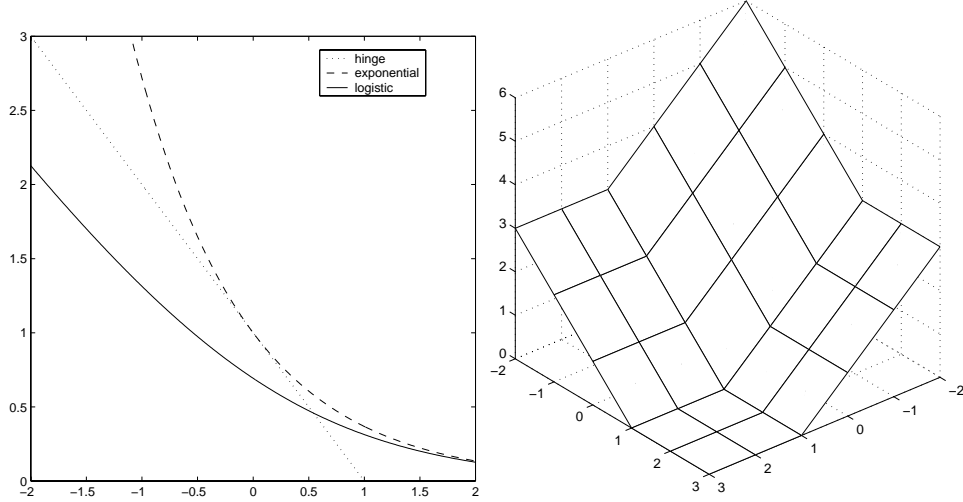

Figure 1: Margin maximizing loss functions for 2-class problems (left) and the SVM 3-class loss function of section 4.1 (right)

The $l_p$-regularized problem is now:

$$(9) \quad \hat{\beta}(\lambda) = \arg \min_{\beta^{(1)},...,\beta^{(K)}} \sum_i C(y_i, h(\mathbf{x}_i)'\beta^{(1)}, ..., h(\mathbf{x}_i)'\beta^{(K)}) + \lambda \sum_k \|\beta^{(k)}\|_p^p$$

Where $\hat{\beta}(\lambda) = (\hat{\beta}^{(1)}(\lambda), ..., \hat{\beta}^{(K)}(\lambda))' \in \mathcal{R}^{K \cdot |\mathcal{H}|}$.

In this formulation, the concept of margin maximization corresponds to maximizing the minimal of all $n \cdot (K-1)$ normalized $l_p$-margins generated by the data:

$$(10) \quad \max_{\|\beta^{(1)}\|_p^p + ... + \|\beta^{(K)}\|_p^p = 1} \min_i \min_{y_i \neq c_k} h(\mathbf{x}_i)'(\beta^{(y_i)} - \beta^{(k)})$$

Note that this margin maximization problem still has a natural geometric interpretation, as $h(\mathbf{x}_i)'(\beta^{(y_i)} - \beta^{(k)}) > 0 \; \forall i, k \neq y_i$ implies that the hyper-plane $h(\mathbf{x})'(\beta^{(j)} - \beta^{(k)}) = 0$ successfully separates classes $j$ and $k$ for any two classes.

Here is a generalization of the optimal separation theorem 2.1 to multi-class models:

**Theorem 4.1** *Assume $C(\mathbf{m})$ is commutative and decreasing in each coordinate, then if $\exists T > 0$ (possibly $T = \infty$ ) such that:*

$$(11) \quad \lim_{t \nearrow T} \frac{C(t[1-\epsilon], tu_1, ...tu_{K-2})}{C(t, tv_1, ..., tv_{K-2})} = \infty,$$
$$\forall \epsilon > 0, \; u_1 \geq 1, ..., u_{K-2} \geq 1, v_1 \geq 1, ...v_{K-2} \geq 1$$

*Then $C$ is a margin-maximizing loss function for multi-class models, in the sense that any convergence point of the normalized solutions to (9), $\frac{\hat{\beta}(\lambda)}{\|\hat{\beta}(\lambda)\|_p}$, attains the optimal separation as defined in (10)*

**Idea of proof** The proof is essentially identical to the two class case, now considering the $n \cdot (K-1)$ margins on which the loss depends. The condition (11) implies that as the regularization vanishes the model is determined by the minimal margin, and so an optimal model puts the emphasis on maximizing that margin.

**Corollary 4.2** *In the 2-class case, theorem 4.1 reduces to theorem 2.1.*

**Proof** The loss depends on $\beta^{(1)} - \beta^{(2)}$, the penalty on $\|\beta^{(1)}\|_p^p + \|\beta^{(2)}\|_p^p$. An optimal solution to the regularized problem must thus have $\beta^{(1)} + \beta^{(2)} = 0$, since by transforming:

$$\beta^{(1)} \rightarrow \beta^{(1)} - \frac{\beta^{(1)} + \beta^{(2)}}{2} \ , \quad \beta^{(2)} \rightarrow \beta^{(2)} - \frac{\beta^{(1)} + \beta^{(2)}}{2}$$

we are not changing the loss, but reducing the penalty, by Jensen's inequality:

$$\|\beta^{(1)} - \frac{\beta^{(1)} + \beta^{(2)}}{2}\|_p^p + \|\beta^{(2)} - \frac{\beta^{(1)} + \beta^{(2)}}{2}\|_p^p = 2\|\frac{\beta^{(1)} - \beta^{(2)}}{2}\|_p^p \le \|\beta^{(1)}\|_p^p + \|\beta^{(2)}\|_p^p$$

So we can conclude that $\hat{\beta}^{(1)}(\lambda) = -\hat{\beta}^{(2)}(\lambda)$ and consequently that the two margin maximization tasks (2), (10) are equivalent.

### 4.1   Margin maximization in multi-class SVM and logistic regression

Here we apply theorem 4.1 to versions of multi-class logistic regression and SVM.

For logistic regression, we use a slightly different formulation than the "standard" logistic regression models, which uses class $K$ as a "reference" class, i.e. assumes that $\beta^{(K)} = 0$. This is required for non-regularized fitting, since without it the solution is not uniquely defined. However, using regularization as in (9) guarantees that the solution will be unique and consequently we can "symmetrize" the model — which allows us to apply theorem 4.1. So the loss function we use is (assume $y = c_k$ belongs to class $k$):

$$(12) \quad C(y, f_1, ..., f_K) \quad = -\log \frac{e^{f_k}}{e^{f_1} + ... + e^{f_K}} =$$
$$= \log(e^{f_1 - f_k} + ... + e^{f_{k-1} - f_k} + 1 + e^{f_{k+1} - f_k} + ... + e^{f_K - f_k})$$

with the linear model: $f_j(\mathbf{x}_i) = h(\mathbf{x}_i)'\beta^{(j)}$. It is not difficult to verify that condition (11) holds for this loss function with $T = \infty$, using the fact that $log(1 + \epsilon) = \epsilon + O(\epsilon^2)$. The sum of exponentials which results from applying this first-order approximation satisfies (11), and as $\epsilon \rightarrow 0$, the second order term can be ignored.

For support vector machines, consider a multi-class loss which is a natural generalization of the two-class loss:

$$(13) \quad\quad\quad\quad\quad\quad C(\mathbf{m}) = \sum_{j=1}^{K-1} [1 - m_j]_+$$

Where $m_j$ is the j'th component of the multi-margin $\mathbf{m}$ as in (8). Figure 1 shows this loss for $K = 3$ classes as a function of the two margins. The loss+penalty formulation using 13 is equivalent to a standard optimization formulation of multi-class SVM (e.g. [11]):

$$\begin{aligned}
\max \quad & c \\
\text{s.t.} \quad & h(\mathbf{x}_i)'(\beta^{(y_i)} - \beta^{(k)}) \ge c(1 - \xi_{ik}), \ \ i \in \{1, ...n\}, \ k \in \{1, ..., K\}, \ c_k \ne y_i \\
& \xi_{ik} \ge 0 \ , \quad \sum_{i,k} \xi_{ik} \le B \ , \quad \sum_k \|\beta^{(k)}\|_p^p = 1
\end{aligned}$$

As both theorem 4.1 (using $T = 1$) and the optimization formulation indicate, the regularized solutions to this problem converge to the $l_p$ margin maximizing multi-class solution.

## 5 Discussion

What are the properties we would like to have in a classification loss function? Recently there has been a lot of interest in Bayes-consistency of loss functions and algorithms ([1] and references therein), as the data size increases. It turns out that practically all "reasonable" loss functions are consistent in that sense, although convergence rates and other measures of "degree of consistency" may vary.

Margin maximization, on the other hand, is a finite sample optimality property of loss functions, which is potentially of decreasing interest as sample size grows, since the training data-set is less likely to be separable. Note, however, that in very high dimensional predictor spaces, such as those typically used by boosting or kernel SVM, separability of any finite-size data-set is a mild assumption, which is violated only in pathological cases.

We have shown that the margin maximizing property is shared by some popular loss functions used in logistic regression, support vector machines and boosting. Knowing that these algorithms "converge", as regularization vanishes, to the same model (provided they use the same regularization) is an interesting insight. So, for example, we can conclude that 1-norm support vector machines, exponential boosting and $l_1$-regularized logistic regression all facilitate the *same* non-regularized solution, which is an $l_1$-margin maximizing separating hyper-plane. From Mangasarian's theorem [7] we know that this hyper-plane maximizes the $l_\infty$ distance from the closest points on either side.

The most interesting statistical question which arises is: *are these "optimal" separating models really good for prediction*, or should we expect regularized models to always do better in practice? Statistical intuition supports the latter, as do some margin-maximizing experiments by Breiman [2] and Grove and Schuurmans [5]. However it has also been observed that in many cases margin-maximization leads to reasonable prediction models, and does not necessarily result in over-fitting. We have had similar experience with boosting and kernel SVM. Settling this issue is an intriguing research topic, and one that is critical in determining the practical importance of our results, as well as that of margin-based generalization error bounds.

## References

[1]  Bartlett, P., Jordan, M. & McAuliffe, J. (2003). Convexity, Classification and Risk Bounds. *Technical reports, dept. of Statistics, UC Berkeley*.

[2]  Breiman, L. (1999). Prediction games and arcing algorithms. *Neural Computation* 7:1493-1517.

[3]  Freund, Y. & Scahpire, R.E. (1995). A decision theoretic generalization of on-line learning and an application to boosting. *Proc. of 2nd Eurpoean Conf. on Computational Learning Theory*.

[4]  Friedman, J. H., Hastie, T. & Tibshirani, R. (2000). Additive logistic regression: a statistical view of boosting. *Annals of Statistics* 28, pp. 337-407.

[5]  Grove, A.J. & Schuurmans, D. (1998). Boosting in the limit: Maximizing the margin of learned ensembles. *Proc. of 15th National Conf. on AI*.

[6]  Hastie, T., Tibshirani, R. & Friedman, J. (2001). *Elements of Stat. Learning*. Springer-Verlag.

[7]  Mangasarian, O.L. (1999). Arbitrary-norm separating plane. *Operations Research Letters, Vol. 24* 1-2:15-23

[8]  Rosset, R., Zhu, J & Hastie, T. (2003). Boosting as a regularized path to a maximum margin classifier. *Technical report, Dept. of Statistics, Stanford Univ.*

[9]  Scahpire, R.E., Freund, Y., Bartlett, P. & Lee, W.S. (1998). Boosting the margin: a new explanation for the effectiveness of voting methods. *Annals of Statistics* 26(5):1651-1686

[10]  Vapnik, V. (1995). *The Nature of Statistical Learning Theory*. Springer.

[11]  Weston, J. & Watkins, C. (1998). Multi-class support vector machines. *Technical report CSD-TR-98-04, dept of CS, Royal Holloway, University of London*.
